# LINEAR LEARNING: LANDSCAPES AND ALGORITHMS

Pierre Baldi
Jet Propulsion Laboratory
California Institute of Technology
Pasadena, CA 91109

What follows extends some of our results of [1] on learning from examples in layered feed-forward networks of linear units. In particular we examine what happens when the number of layers is large or when the connectivity between layers is local and investigate some of the properties of an autoassociative algorithm. Notation will be as in [1] where additional motivations and references can be found. It is usual to criticize linear networks because "linear functions do not compute" and because several layers can always be reduced to one by the proper multiplication of matrices. However this is not the point of view adopted here. It is assumed that the architecture of the network is given (and could perhaps depend on external constraints) and the purpose is to understand what happens during the learning phase, what strategies are adopted by a synaptic weights modifying algorithm,...[see also Cottrell et al. (1988) for an example of an application and the work of Linsker (1988) on the emergence of feature detecting units in linear networks].

Consider first a two layer network with $n$ input units, $n$ output units and $p$ hidden units ($p \leq n$). Let $(x_1, y_1), ..., (x_T, y_T)$ be the set of centered input-output training patterns. The problem is then to find two matrices of weights $A$ and $B$ minimizing the error function $E$:

$$E(A, B) = \sum_{1 \leq t \leq T} ||y_t - ABx_t||^2. \tag{1}$$

Let $\Sigma_{XX}, \Sigma_{XY}, \Sigma_{YY}, \Sigma_{YX}$ denote the usual covariance matrices. The main result of [1] is a description of the landscape of $E$, characerised by a multiplicity of saddle points and an absence of local minima. More precisely, the following four facts are true.

**Fact 1:** For any fixed $n \times p$ matrix $A$ the function $E(A, B)$ is convex in the coefficients of $B$ and attains its minimum for any $B$ satisfying the equation

$$A'AB\Sigma_{XX} = A'\Sigma_{YX}. \tag{2}$$

If in addition $\Sigma_{XX}$ is invertible and $A$ is of full rank $p$, then $E$ is strictly convex and has a unique minimum reached when

$$B = \hat{B}(A) = (A'A)^{-1}A'\Sigma_{YX}\Sigma_{XX}^{-1}. \tag{3}$$

**Fact 2:** For any fixed $p \times n$ matrix $B$ the function $E(A, B)$ is convex in the coefficients of $A$ and attains its minimum for any $A$ satisfying the equation

$$AB\Sigma_{XX}B' = \Sigma_{YX}B'. \tag{4}$$

If in addition $\Sigma_{XX}$ is invertible and $B$ is of full rank $p$, then $E$ is strictly convex and has a unique minimum reached when

$$A = \hat{A}(B) = \Sigma_{YX}B'(B\Sigma_{XX}B')^{-1}. \tag{5}$$

**Fact 3:** Assume that $\Sigma_{XX}$ is invertible. If two matrices $A$ and $B$ define a critical point of $E$ (i.e. a point where $\partial E/\partial a_{ij} = \partial E/\partial b_{ij} = 0$) then the global map $W = AB$ is of the form

$$W = P_A\Sigma_{YX}\Sigma_{XX}^{-1} \tag{6}$$

where $P_A$ denotes the matrix of the orthogonal projection onto the subspace spanned by the columns of $A$ and $A$ satisfies

$$P_A\Sigma = P_A\Sigma P_A = \Sigma P_A \tag{7}$$

with $\Sigma = \Sigma_{YX}\Sigma_{XX}^{-1}\Sigma_{XY}$. If $A$ is of full rank $p$, then $A$ and $B$ define a critical point of $E$ if and only if $A$ satisties (7) and $B = \hat{B}(A)$, or equivalently if and only if $A$ and $W$ satisfy (6) and (7). Notice that in (6), the matrix $\Sigma_{YX}\Sigma_{XX}^{-1}$ is the slope matrix for the ordinary least square regression of $Y$ on $X$.

**Fact 4:** Assume that $\Sigma$ is full rank with $n$ distinct eigenvalues $\lambda_1 > ... > \lambda_n$. If $\mathcal{I} = \{i_1,...,i_p\}(1 \leq i_1 < ... < i_p \leq n)$ is any ordered $p$-index set, let $U_{\mathcal{I}} = [u_{i_1},...,u_{i_p}]$ denote the matrix formed by the orthonormal eigenvectors of $\Sigma$ associated with the eigenvalues $\lambda_{i_1},...,\lambda_{i_p}$. Then two full rank matrices $A$ and $B$ define a critical point of $E$ if and only if there exist an ordered $p$-index set $\mathcal{I}$ and an invertible $p \times p$ matrix $C$ such that

$$A = U_{\mathcal{I}}C \tag{8}$$

$$B = C^{-1}U_{\mathcal{I}}'\Sigma_{YX}\Sigma_{XX}^{-1}. \tag{9}$$

For such a critical point we have

$$W = P_{U_{\mathcal{I}}}\Sigma_{YX}\Sigma_{XX}^{-1} \tag{10}$$

$$E(A, B) = tr(\Sigma_{YY}) - \sum_{i\in\mathcal{I}}\lambda_i. \tag{11}$$

Therefore a critical point of $W$ of rank $p$ is always the product of the ordinary least squares regression matrix followed by an orthogonal projection onto the subspace spanned by $p$ eigenvectors of $\Sigma$. The map $W$ associated with the index set $\{1, 2, ..., p\}$ is the unique local and global minimum of $E$. The remaining $\binom{n}{p} - 1$ $p$-index sets correspond to saddle points. All additional critical points defined by matrices $A$ and $B$ which are not of full rank are also saddle points and can be characerized in terms of orthogonal projections onto subspaces spanned by $q$ eigenvectors with $q < p$.

## Deep Networks

Consider now the case of a deep network with a first layer of $n$ input units, an $(m+1)$-th layer of $n$ output units and $m-1$ hidden layers with an error function given by

$$E(A_1, ..., A_n) = \sum_{1 \leq t \leq T} ||y_t - A_1 A_1 ... A_m x_t||^2. \qquad (12)$$

It is worth noticing that, as in fact 1 and 2 above, if we fix any $m-1$ of the $m$ matrices $A_1, ..., A_m$ then $E$ is convex in the remaining matrix of connection weights. Let $p$ $(p \leq n)$ denote the number of units in the smallest layer of the network (several hidden layers may have $p$ units). In other words the network has a bottleneck of size $p$. Let $i$ be the index of the corresponding layer and set

$$\begin{aligned} A &= A_1 A_2 ... A_{m-i+1} \\ B &= A_{m-i+2} ... A_m \end{aligned} \qquad (13)$$

When we let $A_1, ..., A_m$ vary, the only restriction they impose on $A$ and $B$ is that they be of rank at most $p$. Conversely, any two matrices $A$ and $B$ of rank at most $p$ can always be decomposed (and in many ways) in products of the form of (13). It results that any local minima of the error function of the deep network should yield a local minima for the corresponding "collapsed" three layers network induced by (13) and vice versa. Therefore $E(A_1, ..., A_m)$ does not have any local minima and the global minimal map $W^* = A_1 A_2 ... A_m$ is unique and given by (10) with index set $\{1, 2, ..., p\}$. Notice that of course there is a large number of ways of decomposing $W^*$ into a product of the form $A_1 A_2 ... A_m$. Also a saddle point of the error function $E(A, B)$ does not necessarily generate a saddle point of the corresponding $E(A_1, ..., A_m)$ for the expressions corresponding to the two gradients are very different.

## Forced Connections. Local Connectivity

Assume now an error function of the form

$$E(A) = \sum_{1 \leq t \leq T} ||y_t - Ax_t||^2 \tag{14}$$

for a two layers network but where the value of some of the entries of $A$ may be externally prescribed. In particular this includes the case of local connectivity described by relations of the form $a_{ij} = 0$ for any output unit $i$ and any input unit $j$ which are not connected. Clearly the error function $E(A)$ is convex in $A$. Every constraint of the form $a_{ij}$ =cst defines an hyperplane in the space of all possible $A$. The intersection of all these constraints is therefore a convex set. Thus minimizing $E$ under the given constraints is still a convex optimization problem and so there are no local minima. It should be noticed that, in the case of a network with even only three constrained layers with two matrices $A$ and $B$ and a set of constraints of the form $a_{ij}$ =cst on $A$ and $b_{kl}$ =cst for $B$, the set of admissible matrices of the form $W = AB$ is, in general, not convex anymore. It is not unreasonable to conjecture that local minima may then arise, though this question needs to be investigated in greater detail.

## Algorithmic Aspects

One of the nice features of the error landscapes described so far is the absence of local minima and the existence, up to equivalence, of a unique global minimum which can be understood in terms of principal component analysis and least square regression. However the landscapes are also characterized by a large number of saddle points which could constitute a problem for a simple gradient descent algorithm during the learning phase. The proof in [1] shows that the lower is the $E$ value corresponding to a saddle point, the more difficult it is to escape from it because of a reduction in the possible number of directions of escape (see also [Chauvin, 1989] in a context of Hebbian learning). To assert how relevant these issues are for practical implementations requires further simulation experiments. On a more

speculative side, it remains also to be seen whether, in a problem of large size, the number and spacing of saddle points encountered during the first stages of a descent process could not be used to "get a feeling" for the type of terrain being descented and as a result to adjust the pace (i. e. the learning rate).

We now turn to a simple algorithm for the auto-associative case in a three layers network, i. e. the case where the presence of a teacher can be avoided by setting $y_t = x_t$ and thereby trying to achieve a compression of the input data in the hidden layer. This technique is related to principal component analysis, as described in [1]. If $y_t = x_t$, it is easy to see from equations (8) and (9) that, if we take the matrix $C$ to be the identity, then at the optimum the matrices $A$ and $B$ are transpose of each other. This heuristically suggests a possible fast algorithm for auto-association, where at each iteration a gradient descent step is applied only to one of the connection matrices while the other is updated in a symmetric fashion using transposition and avoiding to back-propagate the error in one of the layers (see [Williams, 1985] for a similar idea). More formally, the algorithm could be concisely described by

$$A(0) = \text{random}$$
$$B(0) = A'(0)$$
$$A(k+1) = A(k) - \eta \frac{\partial E}{\partial A} \qquad (15)$$
$$B(k+1) = A'(k+1)$$

Obviously a similar algorithm can be obtained by setting $B(k+1) = B(k) - \eta \partial E/\partial B$ and $A(k+1) = B'(k+1)$. It may actually even be better to alternate the gradient step, one iteration with respect to $A$ and one iteration with respect to $B$.
A simple calculation shows that (15) can be rewritten as

$$A(k+1) = A(k) + \eta(I - W(k))\Sigma_{XX}A(k)$$
$$B(k+1) = B(k) + \eta B(k)\Sigma_{XX}(I - W(k)) \qquad (16)$$

where $W(k) = A(k)B(k)$. It is natural from what we have already seen to examine the behavior of this algorithm on the eigenvectors of $\Sigma_{XX}$. Assume that $u$ is an eigenvector of both $\Sigma_{XX}$ and $W(k)$ with eigenvalues $\lambda$ and $\mu(k)$. Then it is easy to see that $u$ is an eigenvector of $W(k+1)$ with eigenvalue

$$\mu(k+1) = \mu(k)[1 + \eta\lambda(1 - \mu(k))]^2. \tag{17}$$

For the algorithm to converge to the optimal $W$, $\mu(k+1)$ must converge to 0 or 1. Thus one has to look at the iterates of the function $f(x) = x[1 + \eta\lambda(1 - x)]^2$. This can be done in detail and we shall only describe the main points. First of all, $f'(x) = 0$ iff $x = 0$ or $x = x_a = 1 + (1/\eta\lambda)$ or $x = x_b = 1/3 + (1/3\eta\lambda)$ and $f''(x) = 0$ iff $x = x_c = 2/3 + (2/3\eta\lambda) = 2x_b$. For the fixed points, $f(x) = x$ iff $x = 0$, $x = 1$ or $x = x_d = 1 + (2/\eta\lambda)$. Notice also that $f(x_a) = 0$ and $f(1 + (l/\eta\lambda)) = (1 + (l/\eta\lambda)(1 - l)^2$. Points corresponding to the values $0, 1, x_a, x_d$ of the $x$ variable can readily be positioned on the curve of $f$ but the relative position of $x_b$ (and $x_c$) depends on the value assumed by $\eta\lambda$ with respect to $1/2$. Obviously if $\mu(0) = 0, 1$ or $x_d$ then $\mu(k) = 0, 1$ or $x_d$, if $\mu(0) < 0$ $\mu(k) \to -\infty$ and if $\mu(k) > x_d$ $\mu(k) \to +\infty$. Therefore the algorithm can converge only for $0 \leq \mu(0) \leq x_d$. When the learning rate is too large, i. e. when $\eta\lambda > 1/2$ then even if $\mu(0)$ is in the interval $(0, x_d)$ one can see that the algorithm does not converge and may even exhibit complex oscillatory behavior. However when $\eta\lambda < 1/2$, if $0 < \mu(0) < x_a$ then $\mu(k) \to 1$, if $\mu(0) = x_a$ then $\mu(k) = 0$ and if $x_a < \mu(0) < x_d$ then $\mu(k) \to 1$.

In conclusion, we see that if the algorithm is to be tested, the learning rate should be chosen so that it does not exceed $1/2\lambda$, where $\lambda$ is the largest eigenvalue of $\Sigma_{XX}$. Even more so than back propagation, it can encounter problems in the proximity of saddle points. Once a non-principal eigenvector of $\Sigma_{XX}$ is learnt, the algorithm rapidly incorporates a projection along that direction which cannot be escaped at later stages. Simulations are required to examine the effects of "noisy gradients" (computed after the presentation of only a few training examples), multiple starting points, variable learning rates, momentum terms, and so forth.

## Aknowledgement

Work supported by NSF grant DMS-8800323 and in part by ONR contract 411P006—01.

## References

(1) Baldi, P. and Hornik, K. (1988) Neural Networks and Principal Component Analysis: Learning from Examples without Local Minima. Neural Networks, Vol. **2**, No. 1.

(2) Chauvin, Y. (1989) Another Neural Model as a Principal Component Analyzer. Submitted for publication.

(3) Cottrell, G. W., Munro, P. W. and Zipser, D. (1988) Image Compression by Back Propagation: a Demonstration of Extensional Programming. In: Advances in Cognitive Science, Vol. **2**, Sharkey, N. E. ed., Norwood, NJ Abbex.

(4) Linsker, R. (1988) Self-Organization in a Perceptual Network. Computer 21 (3), 105-117.

(5) Williams, R. J. (1985) Feature Discovery Through Error-Correction Learning. ICS Report 8501, University of California, San Diego.